# Unsupervised Learning in Neurodynamics Using the Phase Velocity Field Approach

**Michail Zak** **Nikzad Toomarian**

Center for Space Microelectronics Technology
Jet Propulsion Laboratory
California Institute of Technology
Pasadena, CA 91109

## ABSTRACT

A new concept for unsupervised learning based upon examples introduced to the neural network is proposed. Each example is considered as an interpolation node of the velocity field in the phase space. The velocities at these nodes are selected such that all the streamlines converge to an attracting set imbedded in the subspace occupied by the cluster of examples. The synaptic interconnections are found from learning procedure providing selected field. The theory is illustrated by examples.

This paper is devoted to development of a new concept for unsupervised learning based upon examples introduced to an artificial neural network. The neural network is considered as an adaptive nonlinear dissipative dynamical system described by the following coupled differential equations:

$$\dot{u}_i + \kappa u_i = \sum_{j=1}^{N} T_{ij} g(u_j) + I_i \qquad i = 1, 2, \ldots, N \qquad (1)$$

in which $u$ is an $N$-dimensional vector, function of time, representing the neuron activity, $T$ is a constant matrix whose elements represent synaptic interconnections between the neurons, $g$ is a monotonic nonlinear function, $I_i$ is the constant exterior input to each neuron, and $\kappa$ is a positive constant.

Let us consider a pattern vector $\tilde{u}$ represented by its end point in an $n$-dimensional phase space, and suppose that this pattern is introduced to the neural net in the form of a set of vectors - examples $u^{(k)}, k = 1, 2 \ldots K$ (Fig. 1). The difference between these examples which represent the same pattern can be caused not only by noisy measurements, but also by the invariance of the pattern to some changes in the vector coordinates (for instance, to translations, rotations etc.). If the set of the points $u^{(k)}$ is sufficiently dense, it can be considered as a finite-dimensional approximation of some subspace $\theta^{(\ell)}$.

Now the goal of this study is formulated as following: find the synaptic interconnections $T_{ij}$ and the input to the network $I_i$ such that any trajectory which is originated inside of $\theta^{(\ell)}$ will be entrapped there. In such a performance the subspace $\theta^{(\ell)}$ practically plays the role of the basin of attraction to the original pattern $\tilde{u}$. However, the position of the attractor itself is not known in advance: the neural net has to create it based upon the introduced representative examples. Moreover, in general the attractor is not necessarily static: it can be periodic, or even chaotic.

The achievement of the goal formulated above would allow one to incorporate into a neural net a set of attractors representing the corresponding clusters of patterns, where each cluster is imbedded into the basin of its attractor. Any new pattern introduced to such a neural net will be attracted to the "closest" attractor. Hence, the neural net would learn by examples to perform content-addressable memory and pattern recognition.

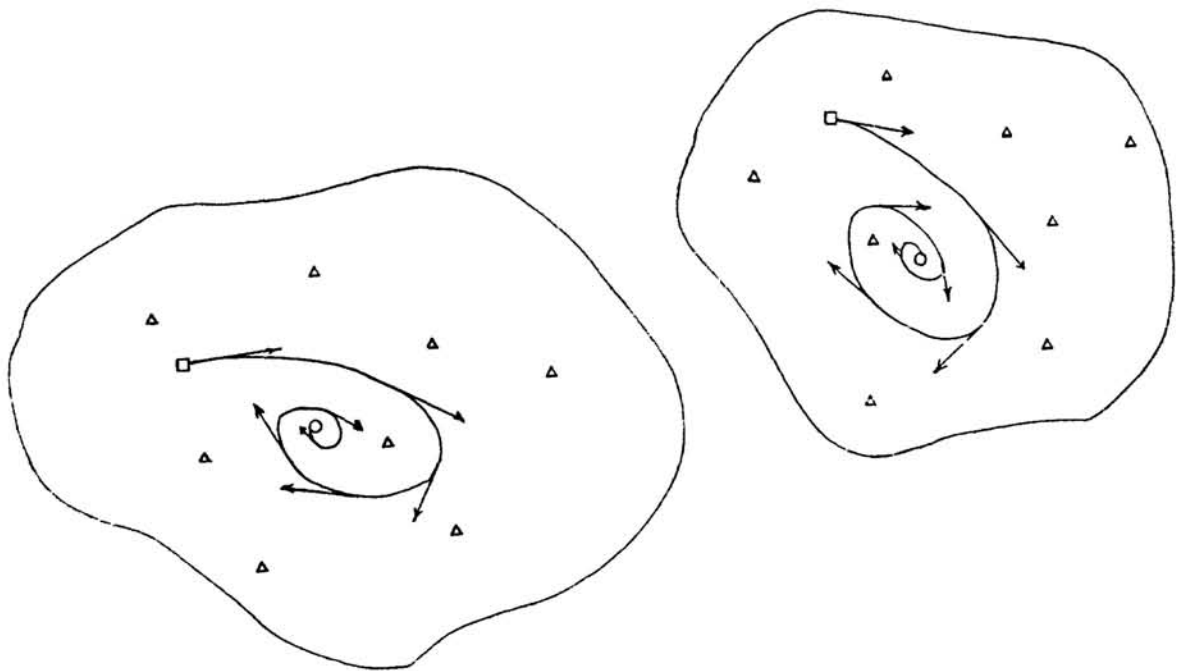

Fig. 1: Two-Dimensional Vectors as Examples, $u^k$, and Formation of Clusters $\theta$.

Our approach is based upon the utilization of the original clusters of the example points $u^{(k)}$ as interpolation nodes of the velocity field in the phase space. The assignment of a certain velocity to an example point imposes a corresponding constraint upon the synaptic interconnections $T_{ij}$ and the input $I_i$ via Eq. (1). After these unknowns are found, the velocity field in the phase space is determined by Eq. (1). Hence, the main problem is to assign velocities at the point examples such that the required dynamical behavior of the trajectories formulated above is provided.

One possibility for the velocity selection based upon the geometrical center approach was analyzed by M. Zak, (1989). In this paper a "gravitational attraction" approach to the same problem will be introduced and discussed.

Suppose that each example-point $u^{(k)}$ is attracted to all the other points $u^{(k')}(k' \neq k)$ such that its velocity is found by the same rule as a gravitational force:

$$v_i^{(k)} = v_o \sum_{\substack{k'=1 \\ k' \neq k}}^{K} \frac{u_i^{(k')} - u_i^{(k)}}{\left[ \sum_{j=1}^{n} (u_j^{(k')} - u_j^{(k)})^2 \right]^{3/2}} \tag{2}$$

in which $v_o$ is a constant scale coefficient.

Actual velocities at the same points are defined by Eq. (1) rearranged as:

$$\dot{u}_i^{(k)} = \sum_{j=1}^{N} T_{ij} g(u_j^{(\kappa)} - u_{oi}) - \kappa(u_i^{(k)} - u_{oi}) \quad \begin{array}{l} i = 1, 2, \ldots, N \\ k = 1, 2, \ldots, K \end{array} \tag{3}$$

The objective is to find synaptic interconnections $T_{ij}$ and center of gravity $u_{oi}$ such that they minimize the distance between the assigned velocity (Eq. 2) and actual calculated velocities (Eq. 3).

Introducing the energy:

$$E = \frac{1}{2} \sum_{k=1}^{K} \sum_{i=1}^{N} (v_i^{(k)} - \dot{u}_i^{(k)})^2 \tag{4}$$

one can find $T_{ij}$ and $u_{oi}$ from the condition:

$$E \to \min$$

i.e., as the static attractor of the dynamical system:

$$\dot{u}_{oi} = -\alpha^2 \frac{\partial E}{\partial u_{oi}} \tag{5a}$$

$$\dot{T}_{ij} = -\alpha^2 \frac{\partial E}{\partial T_{ij}} \tag{5b}$$

in which $\alpha$ is a time scale parameter for learning. By appropriate selection of this parameter the convergence of the dynamical system can be considerably improved (J. Barhen, S. Gulati, and M. Zak, 1989).

Obviously, the static attractor of Eqs. (5) is unique. As follows from Eq. (3)

$$\frac{\partial \dot{u}_i^{(k)}}{\partial u_j^{(k)}} = T_{ij} \frac{dg_j^{(k)}}{du_j^{(k)}}. \qquad (i \neq j) \tag{6}$$

Since $g(u)$ is a monotonic function, $\mathrm{sgn} \frac{dg_j^{(k)}}{du_j^{(k)}}$ is constant which in turn implies that

$$\mathrm{sgn} \frac{\partial \dot{u}_i^{(k)}}{\partial u_j^{(k)}} = \mathrm{const} \quad (i \neq j) \tag{7}$$

Applying this result to the boundary of the cluster one concludes that the velocity at the boundary is directed inside of the cluster (Fig. 2).

For numerical illustration of the new learning concept developed above, we select 6 points in the two dimensional space, (i.e., two neurons) which constructs two separated clusters (Fig. 3, points 1-3 and 16-18 (three points are the minimum to form a cluster in two dimensional space)). Coordinates of the points in Fig. 3 are given in Table 1. The assigned velocity $v_i^k$ calculated based on Eq. 2 and $v_o = 0.04$ are shown in dotted line. For a random initialization of $T_{ij}$ and $u_{oi}$, the energy decreases sharply from an initial value of 10.608 to less than 0.04 in about 400 iterations and at about 2000 iterations the final value of 0.0328 has been achieved, (Fig. 4). To carry out numerical integration of the differential equations, first order Euler numerical scheme with time step of 0.01 has been used. In this simulation the scale parameter $\alpha^2$ was kept constant and set to one. By substituting the calculated $T_{ij}$ and $u_{oi}$ into Eq. (3) for point $u^k$, $(k = 1, 2, 3, 16, 17, 18)$, one will obtain the calculated velocities at these points (shown as dashed lines in Fig. 3). As one may notice, the assigned and calculated velocities are not exactly the same. However, this small difference between the velocities are of no importance as long as the calculated velocities are directed toward the interior of the cluster. This directional difference of the velocities is one of the reasons that the energy did not vanish. The other reason is the difference in the value of these velocities, which is of no importance either, based on the concept developed.

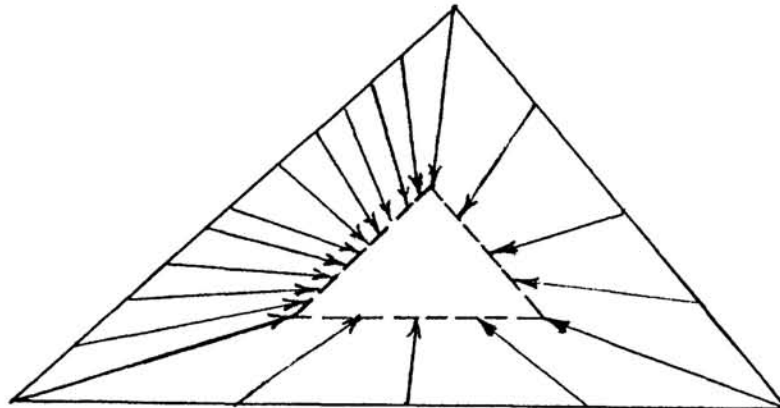

Fig. 2: Velocities at Boundaries are directed Toward Inside of the Cluster.

In order to show that for different initial conditions, Eq. 3 will converge to an attractor which is inside one of the two clusters, this equation was started from different points (4-15, 19-29). In all points, the equation converges to either (0.709,0.0) or (-0.709,0.0). However, the line $x = 0$ in this case is the dividing line, and all the points on this line will converge to $u_o$.

The decay coefficient $\kappa$ and the gain of the hyperbolic tangent were chosen to be 1. However, during the course of this simulation it was observed that the system is very sensitive to these parameters as well as $v_o$, which calls for further study in this area.

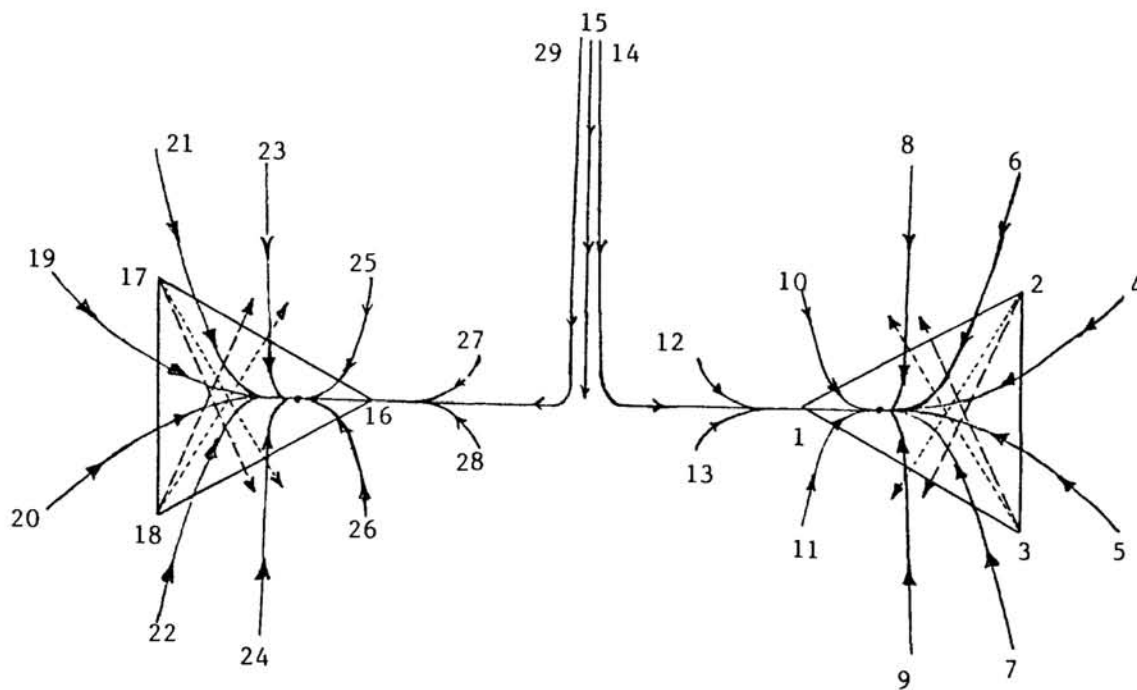

Fig. 3: •  Cluster 1 (1-3) and Cluster 2 (16-19).

• Assigned Velocity (· ·)     Calculated Velocity (- -)

• Activation Dynamics initiated at different points.

Table 1. - Coordinate of Points in Figure 4.

| point | X | Y | point | X | Y |
|---|---|---|---|---|---|
| 1 | 0.50 | 0.00 | 16 | -0.50 | 0.00 |
| 2 | 1.00 | 0.25 | 17 | -1.00 | 0.25 |
| 3 | 1.00 | -0.25 | 18 | -1.00 | 0.25 |
| 4 | 1.25 | 0.25 | 19 | -1.25 | 0.25 |
| 5 | 1.25 | -0.25 | 20 | -1.25 | -0.25 |
| 6 | 1.00 | 0.50 | 21 | -1.00 | 0.50 |
| 7 | 1.00 | -0.50 | 22 | -1.00 | -0.50 |
| 8 | 0.75 | 0.50 | 23 | -0.75 | 0.50 |
| 9 | 0.75 | -0.50 | 24 | -0.75 | -0.50 |
| 10 | 0.50 | 0.25 | 25 | -0.50 | -0.25 |
| 11 | 0.50 | -0.25 | 26 | -0.50 | -0.25 |
| 12 | 0.25 | 0.10 | 27 | -0.25 | 0.10 |
| 13 | 0.25 | -0.10 | 28 | -0.25 | -0.10 |
| 14 | 0.02 | 1.00 | 29 | -0.02 | 1.00 |
| 15 | 0.00 | 1.00 | | | |

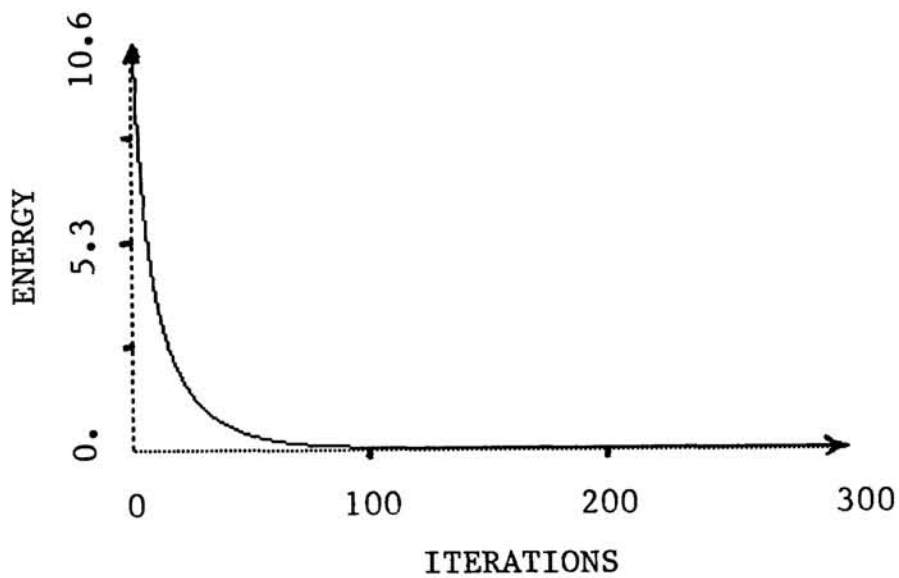

Fig 4: Profile of Neuromorphic Energy over Time Iterations

## Acknowledgement

This research was carried out at the Center for Space Microelectronic Technology, Jet Propulsion Laboratory, California Institute of Technology. Support for the work came from Agencies of the U.S. Department of Defense, including the Innovative Science and Technology Office of the Strategic Defense Initiative Organization and the Office of the Basic Energy Sciences of the US Dept. of Energy, through an agreement with the National Aeronautics and Space Administration.

References

M. Zak (1989), "Unsupervised Learning in Neurondynamics Using Example Interaction Approach", Appl. Math. Letters, Vol. 2, No. 3, pp. 381- 286.

J. Barhen, S. Gulati, M. Zak (1989), "Neural Learning of Constrained nonlinear Transformations", IEEE Computer, Vol. 22(6), pp. 67-76.